# Empowering Visible-Infrared Person Re-Identification with Large Foundation Models

**Zhangyi Hu**[1*]    **Bin Yang**[1*]    **Mang Ye**[1†]

[1]National Engineering Research Center for Multimedia Software,
School of Computer Science, Wuhan University, Wuhan, China.
{zhangyi_hu,yangbin_cv,yemang}@whu.edu.cn
https://github.com/WHU-HZY/TVI-LFM

## Abstract

Visible-Infrared Person Re-identification (VI-ReID) is a challenging cross-modal retrieval task due to significant modality differences, primarily resulting from the absence of color information in the infrared modality. The development of large foundation models like Large Language Models (LLMs) and Vision Language Models (VLMs) motivates us to explore a feasible solution to empower VI-ReID with off-the-shelf large foundation models. To this end, we propose a novel Text-enhanced VI-ReID framework driven by Large Foundation Models (TVI-LFM). The core idea is to enrich the representation of the infrared modality with textual descriptions automatically generated by VLMs. Specifically, we incorporate a pre-trained VLM to extract textual features from texts generated by VLM and augmented by LLM, and incrementally fine-tune the text encoder to minimize the domain gap between generated texts and original visual modalities. Meanwhile, to enhance the infrared modality with extracted textual representations, we leverage modality alignment capabilities of VLMs and VLM-generated feature-level filters. This enables the text model to learn complementary features from the infrared modality, ensuring the semantic structural consistency between the fusion modality and the visible modality. Furthermore, we introduce modality joint learning to align features across all modalities, ensuring that textual features maintain stable semantic representation of overall pedestrian appearance during complementary information learning. Additionally, a modality ensemble retrieval strategy is proposed to leverage complementary strengths of each query modality to improve retrieval effectiveness and robustness. Extensive experiments on three expanded VI-ReID datasets demonstrate that our method significantly improves the retrieval performance, paving the way for the utilization of large foundation models in downstream multi-modal retrieval tasks.

## 1  Introduction

Person Re-Identification (ReID) aims to retrieve images of the same identity across different cameras, which is an important task for intelligent surveillance and urban security [14, 62]. Although RGB-based methods [22, 16, 8, 23, 33, 25, 69, 55, 47, 63, 68, 13] have shown promising results during the daytime, their performance significantly declines at night, as RGB cameras fail to capture sufficient information about individuals in low-light conditions. Infrared cameras can capture pedestrian appearances in dark environments. Therefore, Visible-Infrared Person Re-Identification (VI-ReID) is proposed to match images of individuals captured by visible and infrared cameras, enabling 24-hour surveillance. However, the absence of crucial information, such as color, in infrared images results in significant differences between infrared and visible modalities, posing a major challenge for VI-ReID.

---

[*]Equal Contribution.
[†]Corresponding Author

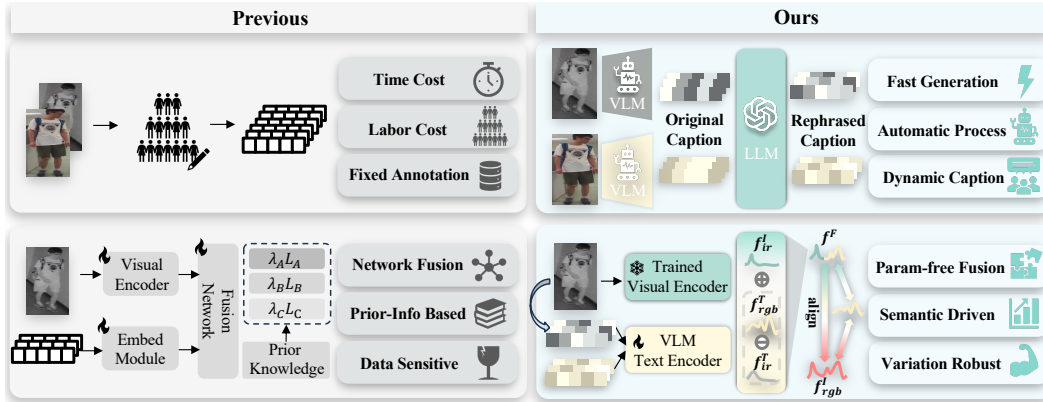

Figure 1: Illustration of our idea. Existing methods rely on manual annotations, complex architectures and prior-knowledge-based optimization to enrich infrared modality, leading to significant time and labor cost, additional parameters and data sensitivity. In contrast, our method employs VLMs and LLMs to automatically generate dynamic text, enhancing the robustness against text variation. Addtionally, we fine-tune a pre-trained VLM through aligning features across all modalities, enabling the framework to create fusion features semantically consistent with visible modality in a parameter-free manner.

Most existing VI-ReID methods, including supervised methods [59, 57, 20, 56, 28, 58, 11, 53, 60], semi-supervised [39, 45] methods and unsupervised methods [51, 38, 37, 49, 50, 52, 48, 61], primarily focus on mining modality-shared features, while paying less attention to compensating for information absence in the infrared modality, which limits further improvements in cross-modal retrieval performance.

In real-world scenarios, based on the visible modality, human descriptions can provide rich information, such as color, serving as vital auxiliary clues. However, existing methods that utilize auxiliary text [9] to enhance the infrared modality, as shown in Fig. 1, heavily rely on human annotation to collect fixed text descriptions, resulting in significant time and labor costs. Moreover, they depend on prior knowledge, such as pre-defined color vocabularies or hyper-parameters, to design complex loss functions and modules with additional parameters for modality alignment and fusion. This reliance increases sensitivity to data variations and reduces the effectiveness of the fusion process.

Recent advancements in large foundation models [2], particularly LLMs and VLMs, demonstrate significant potential for multi-modal retrieval tasks. This motivates us to explore a feasible solution to complement missing vital information with off-the-shelf large foundation models. To this end, we propose a Text-enhanced VI-ReID framework driven by Large Foundation Models (TVI-LFM) which comprises Modality-Specific Caption (MSC), Incremental Fine-tuning Strategy (IFS), and Modality Ensemble Retrieval (MER). The core idea is to enrich infrared representations with generated text, which is a cross-modality retrieval approach bolstered by heterogeneous text descriptions. Specifically, the MSC employs an LLM to augment VLM-generated texts, creating dynamic descriptions of visible and infrared images, reducing labor and time costs, while enhancing the model's robustness against text variations. Then, IFS incorporates a pre-trained VLM to extract features from text generated by MSC, and incrementally fine-tunes the text encoder to minimize the domain gap between the generated texts and the original visual modalities. To enhance the infrared modality with extracted textual representations, IFS leverages the modality alignment capabilities of VLMs and VLM-generated feature-level filters to create a fusion modality. This enables the text model to learn complementary features from the infrared modality, ensuring semantic structural consistency between the fusion modality and the visible modality. Furthermore, IFS introduces modality joint learning to align features across all modalities, ensuring that textual features maintain a stable semantic representation of the overall pedestrian appearance during complementary information learning. Additionally, MER is introduced to leverage complementary strengths of each query modality, forming ensemble queries to further improve retrieval performance. Finally, by integrating the above three modules for dynamic text generation, semantic alignment, and the integration of complementary queries, our method effectively addresses the information absence in the infrared modality, significantly improving cross-modal retrieval performance.

The main contributions can be summarized as follows:

- We design a Text-enhanced VI-ReID framework driven by Large Foundation Models (TVI-LFM). It enriches infrared representations with generated textual descriptions, effectively mitigating the absence of critical information, e.g. color, in the infrared modality and significantly improving the performance of cross-modal retrieval.
- We propose the IFS that fine-tunes a pre-trained VLM to align generated texts with original images. It creates a fusion modality to learn complementary information from the infrared modality and jointly aligns features across all modalities. This ensures a stable semantic consistency of the text, the fusion, and the visible modality during complementary information learning.
- We propose the MER that leverages the complementary strengths of all query modalities, forming ensemble queries to further improve the performance of cross-modality retrieval bolstered by heterogeneous text descriptions.
- We introduce three extended VI-ReID datasets with VLM-generated textual descriptions for every image. Extensive experiments on these expanded datasets demonstrate the competitive performance of our TVI-LFM framework, paving the way for the utilization of large foundation models in downstream multi-modal retrieval tasks.

## 2 Related Work

### 2.1 Visible-Infrared Person Re-Identification

VI-ReID aims to retrieve images across visible and infrared modalities, but it suffers from the absence of critical information, such as color, in infrared modality. Previous methods [59, 4, 64, 27, 57] primarily focus on mining modality-shared information and optimizing features extracted by CNNs or Transformers [54] but pay less attention to compensate for the missing vital information in the infrared modality, which limits further improvements in retrieval performance. Some methods [9, 5, 1] explore auxiliary information compensation. Specifically, [9] uses coarse descriptions as textual identity labels, while [5] and [1] integrate attribute embeddings with visual features. These methods heavily rely on handcrafted annotations for each identity. Furthermore and require prior knowledge, such as hyper-parameters and pre-defined color vocabulary, to design complex modules and loss functions. This results in additional parameters and increased sensitivity to variations in auxiliary data. In contrast, we propose a framework that automatically generates dynamic textual descriptions for VI-ReID datasets and fine-tunes a pre-trained VLM to align the generated texts with the original images. By leveraging modality alignment capabilities and feature-level filters of the VLM, our framework creates a fusion modality that enables the text model to learn complementary features from the infrared modality, while maintaining semantic consistency with the visible modality.

### 2.2 Large Foundation Model

Large foundation models [2], pre-trained on extensive datasets, have shown great potential in downstream tasks. Recent advancements in VLMs like [44, 24, 34, 7, 26] and LLMs such as [35, 3, 67, 41, 18], demonstrate remarkable data generation and text-visual alignment capabilities. For instance, BLIP [24] excels at generating textual captions from images, and can be fine-tuned to accommodate various image styles such as infrared images. Vicuna [67], pre-trained on extensive text data, is great at customized text generation and understanding with prompts. CLIP [34]'s pre-training on large-scale image-text pairs enables its basic capability to align text-image semantics. It can also be fine-tuned on downstream cross-modal retrieval tasks [19]. Our approach integrates generative VLMs and LLMs for automatic text generation and dynamic augmentation. And it incorporates a pre-trained VLM into the VI-ReID system to extract features from texts and utilize them to enrich infrared representations.

### 2.3 Multi-modal Analogical Reasoning in VI-ReID

As demonstrated in [30], language analogical reasoning in the language embedding space can be represented using vector arithmetic. For example, the analogy "*man is to woman as king is to* ?" can be solved by finding the word whose embedding vector is closest to:

$$\vec{v}_{\text{"king"}} - \vec{v}_{\text{"man"}} + \vec{v}_{\text{"woman"}}. \tag{1}$$

Subsequently, [21] investigates an identical regularity manifested in the multi-modal vector space. For instance, the relationship can be expressed as:

$$\vec{v}_{\text{image(blue car)}} - \vec{v}_{\text{"blue"}} + \vec{v}_{\text{"red"}} \approx \vec{v}_{\text{image(red car)}}. \tag{2}$$

We establish this property by aligning features across all modalities, subsequently mapping them into a unified embedding space. This enables us to create fusion features in a parameter-free manner.

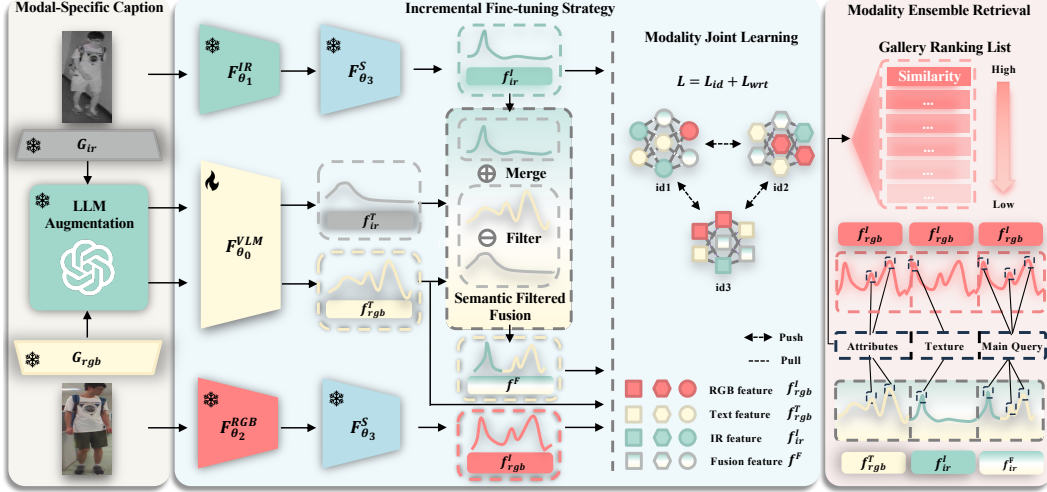

Figure 2: Illustration of our TVI-LFM, including Modality-Specific Caption (MSC), Incremental Fine-tuning Strategy (IFS), and Modality Ensemble Retrieval (MER). MSC utilizes fine-tuned VLMs as modal-specific captioners and employs an LLM for augmentation. IFS fine-tunes a pre-trained VLM to create fusion features semantically consistent with visible features. MER leverages the strengths of all query modalities, forming ensemble queries to improve the retrieval performance.

## 3 Proposed Method

**Task Setting.** Humans can provide textual descriptions based on visible modality. Containing critical information, such as colors, they can serve as auxiliary clues to identify individuals. Therefore, we propose the cross-modal retrieval bolstered by heterogeneous text descriptions. Let $D = \{D_{rgb}^I, D_{ir}^I, Y\}$ denote the sample set, where $D_{rgb}^I = \{V_{rgb}^i\}_{i=1}^{N_{rgb}}$ denotes $N_{rgb}$ visible images, $D_{ir}^I = \{V_{ir}^i\}_{i=1}^{N_{ir}}$ represents $N_{ir}$ infrared images, and $Y = \{y_i\}_{i=1}^{N_{id}}$ denotes the label set. During inference, the $i$-th query sample $q_i$ consists of an infrared image $V_{ir}^i$, a text description $T_{ir}^i$ generated from $V_{ir}^i$, and a randomly selected text $T_{rgb}^i$ generated from visible images of the same identity $y_i$, represented as $q_i = \{V_{ir}^i, T_{ir}^i, T_{rgb}^i\}$. The gallery contains visible images represented as $\mathcal{G} = \{V_{rgb}^i\}_{i=1}^{N_g}$. Finally, compute similarity ranking lists between each $q_i$ and $\forall g_j \in \mathcal{G}$ as results.

**Overview.** Detailed in Fig. 2, TVI-LFM contains MSC, IFS and MER. MSC employs two fine-tuned Blips [24] to automatically generate textual descriptions from visible and infrared images , and utilizes LLM for augmentation. IFS trains a VI-ReID backbone to extract vision features, and incrementally fine-tunes a pre-trained CLIP [34] to align generated texts with original images. Then, it creates a fusion modality to learn complementary features from the infrared modality and jointly aligns features across all modalities. It ensures a stable semantic consistency of the text, the fusion and the visible modality during complementary information learning. Additionally, MER leverages the strengths of each query modality, forming ensemble queries for more accurate retrieval.

### 3.1 Modal-Specific Caption (MSC)

MSC utilizes fine-tuned VLMs to automatically generate text from visible and infrared images and employs an off-the-shelf LLM for textual augmentation, consequently creating dynamic descriptions for VI-ReID datasets. This module reduces the time and labor costs of manual annotations while enhancing the system's robustness against the variations in auxiliary text .

**VLM based Textual Generation.** Currently, there are no publicly available large-scale VI-ReID datasets with image-level text annotations. Thus, to reduce labor and time costs, we fine-tune two VLMs to automatically generate text descriptions with critical information, such as color, for every visible and infrared image. Consequently, we construct three expanded datasets: Tri-SYSU-MM01, Tri-LLCM, and Tri-RegDB, each derived from the original datasets [46, 65, 31] respectively.

- **1) RGB Captioner**: First, train a Blip [24] on a large-scale pedestrian image-text dataset [40] as the RGB captioner $G_{rgb}$, to generate text descriptions for each visible image $V_{rgb}^i \in D_{rgb}^I$.
- **2) IR Captioner**: Then, randomly select visible and infrared images pairs in SYSU-MM01's training split for every identity, then apply the RGB captioner in step 1 to generate textual descriptions for

every visible image in these pairs. Then, remove color-related terms from these generated texts by regular expression filters, and build infrared-filtered text pairs dataset with filtered text descriptions and corresponding infrared images in the same expanded visible-infrared image pairs. Finally we fine-tune the Blip [24] in step 1 again on the Infrared-Text (filtered) dataset as the IR Captioner $G_{ir}$, which is able to generate text descriptions without color for each infrared image $V_{ir}^i \in \boldsymbol{D}_{ir}^I$.

• **3) Text Expanding**: Finally, utilize the two refined modality-specific captioners in previous steps to generate text descriptions $T_{rgb}^i = G_{rgb}(V_{rgb}^i)$ and $T_{ir}^i = G_{rgb}(V_{rgb}^i)$ from original images.

The statistics and samples visualization of the expanded datasets Tri-LLCM, Tri-RegDB and Tri-SYSU-MM01 are shown in Appendix A. Through this expansion process, the framework can automatically generate text descriptions for VI-ReID datasets without large-scale manual annotations.

**LLM based Textual Augmentation.** To ensure that the framework can extract robust representations from generated descriptions against text variations while preserving original semantics of sentences, we propose the LLM-based textual augmentation module applied during the training stage. This module regenerates diverse descriptions by rephrasing the original text for the same target. In detail, given an original description, the module employs an LLM to rephrase it, producing an augmented textual description. The LLM is guided by the prompt *"Rephrase the person's description above using similar words. Answer:"*. The augmentation is applied as follows:

$$T_m^{i*} = \begin{cases} llm(T_m^i \mid prompt), & p \\ T_m^i, & 1 - p \end{cases}, \ m \in \{rgb, ir\}, \tag{3}$$

where $T_m^i$ denotes the text descriptions generated from visible or infrared images, $T_m^{i*}$ represents the augmented text $T_m^i$, and $p = 0.5$ reflects that each description variant is equally probable. Benefiting from the powerful prompt-driven text generation capability of LLM, this approach diversifies the textual descriptions while maintaining their original meanings. This forces the model to focus on the core person appearance information, thus enhancing the robustness of our system against text variation. Moreover, we can also apply this augmentation method directly on existing frameworks involving text data processing, without changing the original structure.

### 3.2 Incremental Fine-tuning Strategy (IFS)

IFS incrementally fine-tunes a CLIP [34] based on the frozen visual features extracted by a trained VI-ReID backbone, to minimize the domain gap between the generated texts and original visual modalities, namely, to align the complementary feature across vision and text, thereby creating fusion features semantically consistent with visible modality. The detailed steps are as follow:

**Features Extraction.** During the training stage, each batch contains $N_B$ selected **quads** as inputs, denoted as $\{(V_{ir}^i, V_{rgb}^i, T_{ir}^{i*}, T_{rgb}^{i*})\}_{i=1}^{N_B}$. Here, $T_{ir}^{i*}$ and $T_{rgb}^{i*}$ are augmented texts generated from the $i$-th image pair $V_{ir}^i$ and $V_{rgb}^i$, which is randomly selected belonging to identity $y_i$. IFS first utilizes the Channel Augmentation (CA) [57] strategy to train a dual-stream ResNet-50 [15] as the VI-ReID backbone, detailed in Appendix B. Then utilize the trained backbone to extract infrared features $\boldsymbol{f}_{ir}^{I(i)}$ and visible features $\boldsymbol{f}_{rgb}^{I(i)}$, denoted as:

$$\boldsymbol{f}_{ir}^{I(i)} = F_{\theta_3}^S(F_{\theta_1}^{IR}(V_{ir}^i)) \quad \text{and} \quad \boldsymbol{f}_{rgb}^{I(i)} = F_{\theta_3}^S(F_{\theta_2}^{RGB}(V_{rgb}^i)), \tag{4}$$

where $F_{\theta_1}^{RGB}$ and $F_{\theta_3}^{IR}$ denote the visible/infrared specific layers respectively, $F_{\theta_3}^S$ denotes the shared layers of the visual backbone. Meanwhile, IFS incorporates a CLIP [34] $F_{\theta_0}^{VLM}$ to extract the "text features" $\boldsymbol{f}_{rgb}^{T(i)}$ from visible images texts and "filter features" $\boldsymbol{f}_{ir}^{T(i)}$ from infrared images texts:

$$\boldsymbol{f}_{rgb}^{T(i)} = F_{\theta_0}^{VLM}(T_{rgb}^{i*}) \quad \text{and} \quad \boldsymbol{f}_{ir}^{T(i)} = F_{\theta_0}^{VLM}(T_{ir}^{i*}). \tag{5}$$

**Semantic Filtered Fusion (SFF).** Meanwhile, to enhance the infrared modality with the generated texts, we propose the SFF module that leverages the text-visual alignment capability of VLM and VLM-generated filter features to create fusion features.

Benefiting from the large-scale pre-training on image-text pairs, the CLIP [34] possesses powerful text-visual alignment capability, ensuring that the features extracted from the generated texts can be aligned with the features of original images. Therefore, the text feature $\boldsymbol{f}_{rgb}^{T(i)}$ can work as an alternative for the visible feature $\boldsymbol{f}_{rgb}^{I(i)}$. Similarly, the filter feature $\boldsymbol{f}_{ir}^{T(i)}$ can work as an alternative for

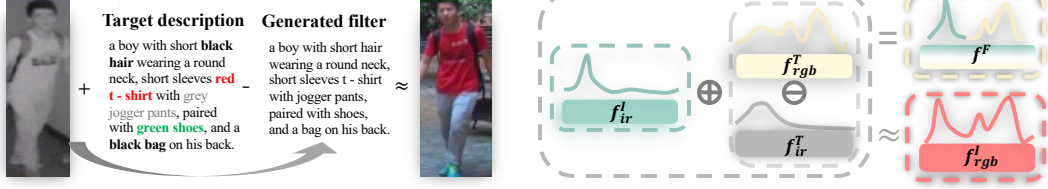

Figure 3: The Visualization of SFF. With the aligned features of generated texts and original images, SFF creates fusion features semantically consistent with visible modality by arithmetically adding the textual complementary information for infrared modality to the infrared features.

the infrared feature $\boldsymbol{f}_{ir}^{I(i)}$. Next, we formulate the complementary features for the infrared modality by decomposing the visible feature $\boldsymbol{f}_{rgb}^{I(i)}$ and the text feature $\boldsymbol{f}_{rgb}^{T(i)}$ as:

$$\boldsymbol{f}_{rgb}^{I(i)} = \boldsymbol{f}_{ir}^{I(i)} + \boldsymbol{f}_{comp}^{I(i)} \quad \text{and} \quad \boldsymbol{f}_{rgb}^{T(i)} = \boldsymbol{f}_{ir}^{T(i)} + \boldsymbol{f}_{comp}^{T(i)}, \tag{6}$$

where $\boldsymbol{f}_{comp}^{I}$ denotes the visual complementary feature for the infrared modality. Similarly, $\boldsymbol{f}_{comp}^{T}$ denotes the textual complementary feature for the infrared modality. Finally, with Eq. (6), and the alignment of features of generated texts and original images, benefiting from the pre-trained CLIP, the representation of fusion features $\boldsymbol{f}^{F}$ with the same semantic structure as the visible modality can be derived, represented as:

$$
\begin{aligned}
\boldsymbol{f}_{rgb}^{I(i)} &= \boldsymbol{f}_{ir}^{I(i)} + \boldsymbol{f}_{comp}^{I(i)} \\
&= \boldsymbol{f}_{ir}^{I(i)} + (\boldsymbol{f}_{rgb}^{I(i)} - \boldsymbol{f}_{ir}^{I(i)}) \\
&\approx \boldsymbol{f}_{ir}^{I(i)} + (\boldsymbol{f}_{rgb}^{T(i)} - \boldsymbol{f}_{ir}^{T(i)}) \cdot \\
&= \boldsymbol{f}_{ir}^{I(i)} + \boldsymbol{f}_{comp}^{T(i)} \\
&\triangleq \boldsymbol{f}^{F(i)}
\end{aligned}
\tag{7}
$$

By leveraging the powerful text-visual alignment capability of CLIP and VLM-generated filter features, SFF approximates $\boldsymbol{f}_{rgb}^{I(i)} - \boldsymbol{f}_{ir}^{I(i)}$ with $\boldsymbol{f}_{rgb}^{T(i)} - \boldsymbol{f}_{ir}^{T(i)}$. Thus, the framework can create fusion features by adding this textual complementary features to the infrared features, as shown in Fig.3. This allows the text model to learn complementary features from the infrared modality, while ensuring semantic structural consistency between the fusion modality and the visible modality.

**Modality Joint Learning (MJL).** Furthermore, MJL is proposed to optimize the pre-trained VLM. It incrementally fine-tunes a CLIP text encoder $F_{\theta_0}^{VLM}$ based on the infrared and visible features extracted from the frozen VI-ReID backbone with parameters $\theta_1, \theta_2$ and $\theta_3$, thus further aligning the fusion modality and the visible modality. This training strategy and its effectiveness in avoiding conflicts during the visual and textual part representation learning are discussed in the Appendix E.

This method utilizes a classic ReID loss for fine-tuning, which eliminates the prior knowledge reliance, such as hyper-parameters and predefined vocabulary, during optimization. The loss consists of a cross-entropy loss $L_{id}$ and a weighted regularized triplet loss $L_{wrt}$ [59], denoted as:

$$L_{total} = L_{id}(B^*; \theta_0) + L_{wrt}(B^*; \theta_0), \quad B^* = \{(\boldsymbol{f}_{rgb}^{T(i)}, \boldsymbol{f}_{rgb}^{I(i)}, \boldsymbol{f}_{ir}^{I(i)}, \boldsymbol{f}^{F(i)}, y_i)\}_{i=1}^{N_B}, \tag{8}$$

where $\theta_0$ denotes the **trainable** parameters of VLM, each batch $B^*$ contains the **frozen** visible feature $\boldsymbol{f}_{rgb}^{I(i)}$, the **frozen** infrared feature $\boldsymbol{f}_{ir}^{I(i)}$, the **trainable** text feature $\boldsymbol{f}_{rgb}^{T(i)}$, the **trainable** fusion feature $\boldsymbol{f}^{F(i)}$ and identity label $y_i$. During optimization, MJL pulls all these features belonging to the same identity $y_i$ together while pushing them away from whom belonging to $\forall y_j \in \{y | y \neq y_i\}$, aligning the semantics across all modalities. Since we solely optimize the $F_{\theta_0}^{VLM}$, this process can be considered as aligning the text features and the fusion features with visible features, under the supervision of an identity label $y_i$ and the regularization from infrared features.

By aligning the textual complementary features for infrared modality $\boldsymbol{f}_{comp}^{T(i)}$ integrated in the fusion features with the corresponding part $\boldsymbol{f}_{comp}^{I(i)}$ in the visible features, MJL effectively reduces the discrepancy between fusion and visible modality. Additionally, it ensures the overall semantic consistency of text features $\boldsymbol{f}_{rgb}^{T(i)}$ with visible features $\boldsymbol{f}_{rgb}^{I(i)}$ during complementary information learning, which enables the following MER to form effective ensemble queries.

Thereby, considering the established two alignments above while taking into account the feature decomposition in Eq. (6), the alignment between infrared features $\boldsymbol{f}_{ir}^{I(i)}$ and filter features $\boldsymbol{f}_{ir}^{T(i)}$ can be derived and represented as:

$$
\begin{aligned}
\boldsymbol{f}_{ir}^{I(i)} &= \boldsymbol{f}_{rgb}^{I(i)} - \boldsymbol{f}_{comp}^{I(i)} \\
&\approx \boldsymbol{f}_{rgb}^{T(i)} - \boldsymbol{f}_{comp}^{T(i)}. \\
&= \boldsymbol{f}_{ir}^{T(i)}
\end{aligned}
\tag{9}
$$

Consequently, by establishing the alignment between generated texts with original images, we successfully minimize the domain gap between the generated texts and original visual modalities, namely, align each part of the visual and textual complementary features, thereby enabling the alignment between fusion features and the visible features, leading to improved retrieval accuracy.

According to Eq. (6), $\boldsymbol{f}_{rgb}^{T(i)}$ can be decomposed as trainable $\boldsymbol{f}_{ir}^{T(i)}$ and $\boldsymbol{f}_{comp}^{T(i)}$, while $\boldsymbol{f}^{F(i)}$ can be decomposed as frozen $\boldsymbol{f}_{ir}^{I(i)}$ and trainable $\boldsymbol{f}_{comp}^{T(i)}$. Considering that frozen infrared features $\boldsymbol{f}_{ir}^{I(i)}$ also participate in aligning with the text features $\boldsymbol{f}_{rgb}^{T(i)}$ and the fusion features $\boldsymbol{f}^{F(i)}$, it can be regarded as a regularization to constrain the complexity of textual complementary features $\boldsymbol{f}_{comp}^{T(i)}$ while aligning with filter features $\boldsymbol{f}_{ir}^{T(i)}$, enabling the framework to construct better fusion features.

### 3.3 Modality Ensemble Retrieval (MER)

MER leverages the complementary modalities strengths and rich semantics in query representations mined from IFS to create an ensemble query feature $\boldsymbol{f}_q^{(i)}$ for more accurate retrieval, represented as:

$$
\boldsymbol{f}_q^{(i)} = (\boldsymbol{f}_{ir}^{I(i)} + \boldsymbol{f}_{rgb}^{T(i)} + \boldsymbol{f}^{F(i)})/3,
\tag{10}
$$

where **fusion feature** $\boldsymbol{f}^{F(i)}$ provides a combined feature with semantic structure the same as the visible features, serving as the primary matching modality. **Infrared feature** $\boldsymbol{f}_{ir}^{I(i)}$ provides contiguous visual semantics. Their similarity with visible images can serve as a supplementary reference for texture and shape information. **Text feature** $\boldsymbol{f}_{rgb}^{T(i)}$, aligned with visible features in MJL, provides descriptive information such as the color and the pattern of the clothes. The similarity between text features and visible features serves as a reference for these key matching features. Therefore, when encountering challenging scenarios that are hard to distinguish a person's clothing or shape but distinguishing color is feasible, or vice versa, the **ensemble query feature** $\boldsymbol{f}_q^{(i)}$ can leverage the features of two modalities in addition to the primary matching contribution from fusion modality to explore their similarities in visual texture and key attributes respectively. Thus, the similarity score $s_{ij}$ between $i$-th query sample and $j$-th gallery sample is defined as $\boldsymbol{f}_q^{(i)} \cdot \boldsymbol{f}_{rgb}^{I(j)}$. Thus, the strengths of all query modalities can be integrated into the final similarity score, consequently enhancing the accuracy against hard cases of retrieval. In fact, by plugging Eq. (10) into $\boldsymbol{f}_q^{(i)} \cdot \boldsymbol{f}_{rgb}^{I(j)}$, an equivalent definition for the similarity between two high-dimension features is derived, represented as:

$$
\begin{aligned}
s_{ij} &= (\boldsymbol{f}_{ir}^{I(i)} + \boldsymbol{f}_{rgb}^{T(i)} + \boldsymbol{f}^{F(i)}) \cdot \boldsymbol{f}_{rgb}^{I(j)}/3 \\
&= concat[\boldsymbol{f}_{ir}^{I(i)}, \boldsymbol{f}_{rgb}^{T(i)}, \boldsymbol{f}^{F(i)}] \cdot concat[\boldsymbol{f}_{rgb}^{I(j)}, \boldsymbol{f}_{rgb}^{I(j)}, \boldsymbol{f}_{rgb}^{I(j)}]/3.
\end{aligned}
\tag{11}
$$

It is equivalent to increasing the feature dimension for retrieval but use ensemble features with less dimensions and computational cost. Due to the larger distances between classes in higher dimensional feature space, the models can distinguish features of different identities in the ensemble feature space more easily, therefore utilizing ensemble queries can further improve the retrieval performance.

## 4 EXPERIMENTS

### 4.1 Experimental Settings

**Datasets.** We evaluate our framework on Tri-SYSU-MM01, Tri-RegDB, and Tri-LLCM. These datasets with text descriptions for each visible and infrared image are expanded from the original VI-ReID datasets [46, 65, 31], utilizing fine-tuned Blip [24] as captioner. The splits of the training set and testing set for each dataset are available in Appendix D.

**Evaluation Protocols.** In line with established VI-ReID settings [59, 57], we assess the performance of the infrared query and the fusion query using Rank-k matching accuracy, mean Average Precision

Table 1: Ablation study on fusion query ($I + T \rightarrow R$) about each component on the performance of **Tri-SYSU-MM01** and **Tri-LLCM** datasets. **Rank-1** (%), **mAP**(%), and **mINP**(%) are reported.

| $I + T \rightarrow R$ | | | | | Tri-SYSU-MM01 | | | Tri-LLCM | | |
|---|---|---|---|---|---|---|---|---|---|---|
| B | SFF | MJL | LLM | MER | R1 | mAP | mINP | R1 | mAP | mINP |
| ✓ | | | | | 72.52 | 69.15 | 55.93 | 52.63 | 58.82 | 55.43 |
| ✓ | ✓ | | | | 77.00 | 73.73 | 61.50 | 54.73 | 60.95 | 57.64 |
| ✓ | ✓ | ✓ | | | 83.97 | 80.40 | 69.46 | 56.76 | 63.58 | 60.35 |
| ✓ | ✓ | ✓ | ✓ | | 84.17 | 80.72 | 70.02 | 57.13 | 64.06 | 60.72 |
| ✓ | ✓ | ✓ | | ✓ | 84.88 | 81.32 | 70.57 | 57.09 | 63.87 | 60.62 |
| ✓ | ✓ | ✓ | ✓ | ✓ | **84.90** | **81.47** | **70.85** | **58.19** | **65.08** | **61.83** |

(mAP), and mean Inverse Negative Penalty (mINP) [59] within our TVI-LFM framework. To get stable performance on Tri-SYSU-MM01 and Tri-LLCM, we evaluate our model 10 times with random splits of the gallery set; as for Tri-RegDB, we evaluate our model on 10 trials with different train/test splits and report the average performance on each dataset.

**Implementation Overview.** We utilize a dual-stream ResNet-50 [58] pre-trained on ImageNet [36] as the visual backbone and a transformer in CLIP [34] as the textual encoder. Training involves visible and infrared images alongside text descriptions generated by two modality-specialized fine-tuned Blip [24] models. All text descriptions are augmented by vicuna-7b [67] with a random rephrasing strategy. Incremental fine-tuning is applied by fixing the visual parameters while tuning the textual part of the framework. All details are described in Appendix B.

## 4.2 Ablation Study

To thoroughly evaluate the effect of each component of our proposed method, we conduct comprehensive ablation studies on Tri-LLCM and Tri-SYSU-MM01. These studies involve gradually adding the proposed modules to our baseline, systematically removing specific modules from our framework and assessing their impact on performance. The overall experimental setup remained consistent, with only the module under evaluation being modified.

**Effect of Semantic Filtered Fusion.** By leveraging text-visual modality alignment capability of VLM and VLM-generated filter features, SFF fuses textual complementary information with infrared features to create fusion features semantically consistent with visible modality. Compared to the baseline, the method obtains a 4.48% Rank-1 improvement in Tri-SYSU-MM01 and a 2.10% Rank-1 improvement in Tri-LLCM, as shown in Table 1. The results demonstrate that the module effectively integrates information from different modalities.

**Effect of Modality Joint Learning.** In cooperation with SFF, MJL aligns features across all modalities to minimize the domain gap between the generated texts and original visual modalities, thereby mitigating the fusion-visible modality difference. Based on the experimental results in Table 1 and compared to the baseline with SFF, adding MJL gains a significant enhancement of 6.97% Rank-1 improvement, 6.67% mAP improvement, and 7.96% mINP improvement in Tri-SYSU-MM01, and 2.03% Rank-1 improvement, 2.63% mAP improvement, and 2.71% mINP improvement in Tri-LLCM. This result demonstrates the effectiveness of MJL, which greatly minimizes the modality gap.

**Effect of Modality Ensemble Retrieval.** MER leverages the complementary advantages of different query modalities to construct ensemble query features thereby improving the retrieval accuracy. The results demonstrate its effectiveness, in Table 1, incorporating MER provides an additional improvement of 0.91% in Rank-1, 0.92% in mAP, and 1.11% in mINP in the Tri-SYSU-MM01 dataset over the baseline with MJL+SFF. Similarly, on the Tri-LLCM dataset, MER achieves 0.33% Rank-1 improvement, 0.29% mAP improvement, and 0.27% mINP improvement.

**Effect of LLM based Textual Augmentation.** To extract robust text representations against text variation, we implement LLM based augmentation module by randomly rephrasing the generated descriptions for dynamic expression. As shown in Table 1, incorporating it further improves the overall performance and robustness against text variation. It also works well with other modules, achieving 84.90% Rank-1 and 58.19% Rank-1 in Tri-SYSU-MM01 and Tri-LLCM respectively.

## 4.3 Comparison with the State-of-the-art Methods

We present a comprehensive comparison of TVI-LFM against state-of-the-art methods on different datasets as outlined in Table 2 and Table 3. Our evaluation includes a variety of metrics: Rank-1

Table 2: Comparison with the state-of-the-art methods on the proposed Tri-SYSU-MM01.

| Methods | Venue | Type | All Search | | | Indoor Search | | |
|---|---|---|---|---|---|---|---|---|
| | | | R-1 | mAP | mINP | R-1 | mAP | mINP |
| Zero-Padding [46] | ICCV-17 | | 14.80 | 15.95 | - | 20.58 | 26.92 | - |
| HCML [56] | AAAI-18 | | 14.32 | 16.16 | - | 24.52 | 30.08 | - |
| cmGAN [6] | IJCAI-18 | | 26.97 | 27.80 | - | 31.63 | 42.19 | - |
| AlignGAN [43] | ICCV-19 | | 42.40 | 40.70 | - | 45.90 | 54.30 | - |
| AGW [59] | TPAMI-21 | | 47.50 | 47.65 | 35.30 | 54.17 | 62.97 | 59.23 |
| DDAG [58] | ECCV-20 | | 54.75 | 53.02 | 39.62 | 61.02 | 67.98 | 62.61 |
| CM-NAS [12] | ICCV-21 | $I \rightarrow R$ | 61.99 | 60.02 | - | 67.01 | 72.95 | - |
| DART [53] | CVPR-22 | | 68.7 | 66.3 | - | 82.0 | 73.8 | - |
| CAJ [57] | ICCV-21 | | 69.88 | 66.89 | 53.61 | 76.26 | 80.37 | 76.79 |
| DEEN [65] | CVPR-23 | | 74.70 | 71.80 | - | 80.30 | 83.30 | - |
| SAAI [10] | ICCV-23 | | 75.90 | 77.03 | - | 83.20 | 88.01 | - |
| MSCLNet [64] | ECCV-22 | | 76.99 | 71.64 | - | 78.49 | 81.17 | - |
| SGIEL [11] | CVPR-23 | | 77.12 | 72.33 | - | 82.07 | 82.95 | - |
| PartMix [20] | CVPR-23 | | 77.78 | 74.62 | - | 81.52 | 84.38 | - |
| YYDS [9] | Arxiv-24 | $I + T \rightarrow R$ | 74.60 | 70.35 | 56.01 | 81.35 | 83.64 | 79.56 |
| VI-ReID Backbone | - | $I \rightarrow R$ | 69.89 | 66.74 | 53.34 | 76.91 | 80.64 | 76.70 |
| **TVI-LFM** | - | $I + T \rightarrow R$ | **84.90** | **81.47** | **70.85** | **89.06** | **90.78** | **88.39** |

Table 3: Comparison with the state-of-the-art methods on the proposed Tri-RegDB and Tri-LLCM.

| Methods | Venue | Type | Tri-RegDB | | | Tri-LLCM | | |
|---|---|---|---|---|---|---|---|---|
| | | | R-1 | mAP | mINP | R-1 | mAP | mINP |
| DDAG [58] | ECCV-20 | | 68.06 | 61.80 | 48.62 | 40.3 | 48.4 | - |
| AGW [59] | TPAMI-21 | | 70.49 | 65.90 | 51.24 | 43.6 | 51.8 | - |
| CAJ [57] | ICCV-21 | $I \rightarrow R$ | 84.8 | 77.8 | 61.56 | 48.8 | 56.6 | - |
| DART [53] | CVPR-22 | | 82.0 | 73.8 | - | 52.2 | 59.8 | - |
| MMN [66] | MM-21 | | 87.5 | 80.5 | - | 52.5 | 58.9 | - |
| DEEN [65] | CVPR-23 | | 89.5 | 83.4 | - | 54.9 | 62.9 | - |
| YYDS [9] | Arxiv-24 | $I + T \rightarrow R$ | 90.95 | 84.22 | 70.12 | 58.13 | 64.91 | 61.77 |
| VI-ReID Backbone | - | $I \rightarrow R$ | 89.51 | 83.51 | 69.65 | 53.53 | 59.77 | 56.40 |
| **TVI-LFM** | - | $I + T \rightarrow R$ | **91.38** | **85.92** | **72.73** | **58.19** | **65.08** | **61.83** |

(R-1), mean Average Precision (mAP), and mean Inverse Negative Penalty (mINP) [59]. For fair comparison, we re-run YYDS on the proposed expanded datasets with the same image size: $288 \times 144$.

**Performance on Tri-SYSU-MM01 Dataset** As shown in Table 2, with the enhancement of generated text, TVI-LFM greatly improves the performance of the VI-ReID backbone and outperforms all previous methods under 'All Search' and 'Indoor Search' conditions. Specifically, TVI-LFM achieves significant improvements in Rank-1, reaching 84.90% and 89.06% respectively, compared to the next best result of 77.78% by PartMix [20] in All Search and 83.20% by SAAI [10] in Indoor Search. Furthermore, in terms of mAP, TVI-LFM posts scores of 81.47% and 90.78%, which are substantial increases from the previous high scores of 77.03% and 88.01%, respectively.

**Performance on Tri-RegDB and Tri-LLCM Dataset** Table 3 outlines our method's performance on the two datasets. In the Tri-RegDB dataset, TVI-LFM obtains a Rank-1 of 91.38% and an mAP of 85.92%, higher than the prior top scores of 90.95% in Rank-1 and 84.22% in mAP by YYDS. In the Tri-LLCM dataset, our method leads with a Rank-1 of 58.19% and an mAP of 65.08%, surpassing the prior top scores of 58.13% in Rank-1 and 64.91% in mAP, both held by YYDS.

## 4.4 Visualization

**Feature Distribution Visualization.** To explore the reason why our method is effective, we utilize t-SNE [42] 2D feature space and visualize cosine distances of the intra-class and inter-class features on Tri-SYSU-MM01 dataset. From the (a) to (d) in Fig. 4, the t-SNE feature distribution shows that our method greatly enhances the ability of distinguishing features from different identities with text and reduces extreme outliers of the same identity and samples with too large cross-modal discrepancy. For the feature distance distribution shown in Fig. 4 (e-h), which corresponds to the 2D t-SNE [42] feature distribution, the inter-class and intra-class distance distributions are increasingly well separated, particularly noting that the excessive intra-class distance is also significantly reduced.

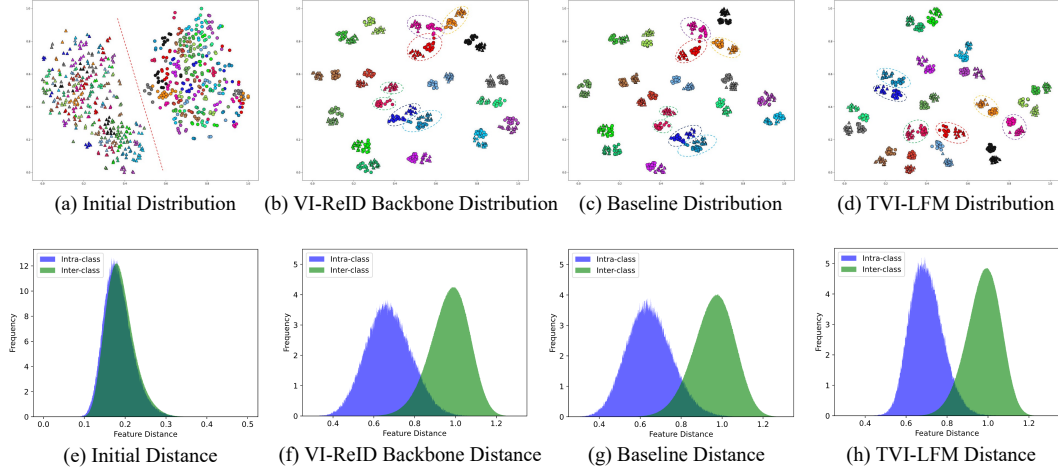

(a) Initial Distribution     (b) VI-ReID Backbone Distribution     (c) Baseline Distribution     (d) TVI-LFM Distribution

(e) Initial Distance     (f) VI-ReID Backbone Distance     (g) Baseline Distance     (h) TVI-LFM Distance

Figure 4: First row (a-d) shows the t-SNE feature distribution of the 20 randomly selected identities, triangles means infrared features (w/wo textual enhancement), circles means visible features. Different colors indicate different identities. Figures in the second row (e-h) represent the intra-class (blue) and inter-class (green) distances of infrared features (w/wo textual fusion) and visible features.

**Retrieval Result.** To intuitively present the performance of our method, we visualize some retrieval results of the VI-ReID backbone, baseline and our method on the Tri-SYSU-MM01 dataset in Appendix C. For the same query image, our method significantly enhances retrieval performance utilizing generated descriptions compared to baseline and VI-ReID backbone.

## 5   Conclusion

To alleviate the absence of detailed color information in the infrared modality, this paper presents a VI-ReID framework driven by Large Foundation Models (TVI-LFM) to enrich the infrared representation with VLM-generated textual descriptions, which is a cross-modality retrieval approach bolstered by heterogeneous text descriptions. To enhance the infrared modality with text, MSC utilizes one off-the-shelf LLM to augment VLM-generated text descriptions. Then, IFS incorporates a pre-trained VLM to extract features from generated texts, and incrementally fine-tunes the text encoder to align generated texts and original visual modalities. To enhance the infrared modality with extracted text features, IFS leverages modality alignment capabilities of VLMs and VLM-generated feature-level filters to create fusion modality. This enables the text model to learn complementary features from the infrared modality, ensuring semantic structural consistency between the fusion modality and the visible modality. Furthermore, IFS introduces modality joint learning to align features of all modalities, maintaining a stable semantic representation of the overall pedestrian appearances for text features during complementary information learning. Additionally, MER leverages complementary strengths of query modalities, forming ensemble queries to further improve retrieval performance. Extensive experiments on three expanded VI-ReID datasets demonstrate the competitive performance of TVI-LFM, paving the way for the utilization of large foundation models in downstream multi-modal retrieval tasks.

**Limitations and future research.** While the proposed TVI-LFM shows promising performance, the retrieval accuracy hinges on text quality, and the performance on hard datasets, such as LLCM[65], still have rooms for improvement. High-quality text enhances retrieval accuracy by improving text-vision correspondences during training and providing precise information for infrared compensation during inference. Therefore, for future improvements, *1) more advanced generative models*; *2) image augmentations during generator fine-tuning*; *3) progressive generation strategies focusing on fine-grained attributes*, could be introduced to enhance text quality, thereby improving the accuracy of cross-modality retrieval bolstered by heterogeneous textual descriptions.

**Acknowledgements.** This work is partially supported by National Natural Science Foundation of China under Grant (62176188, 62361166629, 62225113), Key Research and Development Project of Hubei Province (2022BAD175), and Postdoctoral Fellowship Program of China Postdoctoral Science Foundation (GZC20241268). The numerical calculations in this paper have been done on the supercomputing system in the Supercomputing Center of Wuhan University.

# References

[1] Zheng Aihua, Pan Peng, Li Hongchao, Li Chenglong, Luo Bin, Tan Chang, and Jia Ruoran. Progressive attribute embedding for accurate cross-modality person re-id. In *ACM MM*, 2022.

[2] Rishi Bommasani, Drew A. Hudson, Ehsan Adeli, Russ Altman, Simran Arora, Sydney von Arx, Michael S. Bernstein, Jeannette Bohg, Antoine Bosselut, Emma Brunskill, and et. al. On the opportunities and risks of foundation models. *ArXiv*, 2022.

[3] Tom B. Brown, Benjamin Mann, Nick Ryder, Melanie Subbiah, Jared Kaplan, Prafulla Dhariwal, Arvind Neelakantan, Pranav Shyam, Girish Sastry, Amanda Askell, Sandhini Agarwal, Ariel Herbert-Voss, and et. al. Language models are few-shot learners. In *NeurIPS*, 2020.

[4] Cuiqun Chen, Mang Ye, Meibin Qi, Jingjing Wu, Jianguo Jiang, and Chia-Wen Lin. Structure-aware positional transformer for visible-infrared person re-identification. *IEEE TIP*, 2022.

[5] Zhuxuan Cheng, Huijie Fan, Qiang Wang, Shiben Liu, and Yandong Tang. Dual-stage attribute embedding and modality consistency learning-based visible–infrared person re-identification. *Electronics*, 2023.

[6] Pingyang Dai, Rongrong Ji, Haibin Wang, Qiong Wu, and Yuyu Huang. Cross-modality person re-identification with generative adversarial training. In *IJCAI*, 2018.

[7] Wenliang Dai, Junnan Li, Dongxu Li, Anthony Meng Huat Tiong, Junqi Zhao, Weisheng Wang, Boyang Li, Pascale Fung, and Steven Hoi. Instructblip: Towards general-purpose vision-language models with instruction tuning. *ArXiv*, 2023.

[8] Neng Dong, Shuanglin Yan, Hao Tang, Jinhui Tang, and Liyan Zhang. Multi-view information integration and propagation for occluded person re-identification. *IF*, 2024.

[9] Yunhao Du, Zhicheng Zhao, and Fei Su. Yyds: Visible-infrared person re-identification with coarse descriptions. *ArXiv*, 2024.

[10] Xingye Fang, Yang Yang, and Ying Fu. Visible-infrared person re-identification via semantic alignment and affinity inference. In *ICCV*, 2023.

[11] Jiawei Feng, Ancong Wu, and Wei-Shi Zheng. Shape-erased feature learning for visible-infrared person re-identification. In *CVPR*, 2023.

[12] Chaoyou Fu, Yibo Hu, Xiang Wu, Hailin Shi, Tao Mei, and Ran He. Cm-nas: Cross-modality neural architecture search for visible-infrared person re-identification. In *ICCV*, 2021.

[13] Dengpan Fu, Dongdong Chen, Jianmin Bao, Hao Yang, Lu Yuan, Lei Zhang, Houqiang Li, and Dong Chen. Unsupervised pre-training for person re-identification. In *CVPR*, 2021.

[14] Yunpeng Gong, Liqing Huang, and Lifei Chen. Person re-identification method based on color attack and joint defence. In *CVPR*, 2022.

[15] Kaiming He, Xiangyu Zhang, Shaoqing Ren, and Jian Sun. Deep residual learning for image recognition. In *CVPR*, 2016.

[16] Shuting He, Hao Luo, Pichao Wang, Fan Wang, Hao Li, and Wei Jiang. Transreid: Transformer-based object re-identification. In *ICCV*, 2021.

[17] Andrew G. Howard, Menglong Zhu, Bo Chen, Dmitry Kalenichenko, Weijun Wang, Tobias Weyand, Marco Andreetto, and Hartwig Adam. Mobilenets: Efficient convolutional neural networks for mobile vision applications. *ArXiv*, 2017.

[18] Albert Q. Jiang, Alexandre Sablayrolles, Antoine Roux, Arthur Mensch, Blanche Savary, Chris Bamford, and et. al. Mixtral of experts. *ArXiv*, 2024.

[19] Ding Jiang and Mang Ye. Cross-modal implicit relation reasoning and aligning for text-to-image person retrieval. In *CVPR*, 2023.

[20] Minsu Kim, Seungryong Kim, Jungin Park, Seongheon Park, and Kwanghoon Sohn. Partmix: Regularization strategy to learn part discovery for visible-infrared person re-identification. In *CVPR*, 2023.

[21] Ryan Kiros, Ruslan Salakhutdinov, and Richard S. Zemel. Unifying visual-semantic embeddings with multimodal neural language models. *ArXiv*, 2014.

[22] He Li, Mang Ye, Cong Wang, and Bo Du. Pyramidal transformer with conv-patchify for person re-identification. In *ACM MM*, 2022.

[23] Huafeng Li, Yiwen Chen, Dapeng Tao, Zhengtao Yu, and Guanqiu Qi. Attribute-aligned domain-invariant feature learning for unsupervised domain adaptation person re-identification. *IEEE TIFS*, 2021.

[24] Junnan Li, Dongxu Li, Caiming Xiong, and Steven Hoi. Blip: Bootstrapping language-image pre-training for unified vision-language understanding and generation. *ArXiv*, 2022.

[25] Mingkun Li, Chun-Guang Li, and Jun Guo. Cluster-guided asymmetric contrastive learning for unsupervised person re-identification. *IEEE TIP*, 2022.

[26] Haotian Liu, Chunyuan Li, Qingyang Wu, and Yong Jae Lee. Visual instruction tuning. In *NeurIPS*, 2023.

[27] Jialun Liu, Yifan Sun, Feng Zhu, Hongbin Pei, Yi Yang, and Wenhui Li. Learning memory-augmented unidirectional metrics for cross-modality person re-identification. In *CVPR*, 2022.

[28] Min Liu, Yeqing Sun, Xueping Wang, Yuan Bian, Zhu Zhang, and Yaonan Wang. Pose-guided modality-invariant feature alignment for visible–infrared object re-identification. *IEEE TIM*, 2024.

[29] Hao Luo, Youzhi Gu, Xingyu Liao, Shenqi Lai, and Wei Jiang. Bag of tricks and a strong baseline for deep person re-identification. In *CVPR*, 2019.

[30] Tomas Mikolov, Wen-tau Yih, and Geoffrey Zweig. Linguistic regularities in continuous space word representations. In *ACL*, 2013.

[31] Dat Nguyen, Hyung Hong, Ki Kim, and Kang Park. Person recognition system based on a combination of body images from visible light and thermal cameras. *Sensors*, 2017.

[32] Adam Paszke, Sam Gross, Francisco Massa, Adam Lerer, James Bradbury, Gregory Chanan, Trevor Killeen, Zeming Lin, and et. al. Pytorch: An imperative style, high-performance deep learning library. In *NeurIPS*, 2019.

[33] Xuelin Qian, Yanwei Fu, Tao Xiang, Wenxuan Wang, Jie Qiu, Yang Wu, Yu-Gang Jiang, and X. Xue. Pose-normalized image generation for person re-identification. In *ECCV*, 2017.

[34] Alec Radford, Jong Wook Kim, Chris Hallacy, Aditya Ramesh, Gabriel Goh, Sandhini Agarwal, Girish Sastry, Amanda Askell, Pamela Mishkin, Jack Clark, Gretchen Krueger, and Ilya Sutskever. Learning transferable visual models from natural language supervision. In *ICML*, 2021.

[35] Alec Radford, Jeff Wu, Rewon Child, David Luan, Dario Amodei, and Ilya Sutskever. Language models are unsupervised multitask learners. 2019.

[36] Olga Russakovsky, Jia Deng, Hao Su, Jonathan Krause, Sanjeev Satheesh, Sean Ma, Zhiheng Huang, Andrej Karpathy, Aditya Khosla, Michael S. Bernstein, Alexander C. Berg, and Li Fei-Fei. Imagenet large scale visual recognition challenge. *IJCV*, 2014.

[37] Jiangming Shi, Xiangbo Yin, Yeyun Chen, Yachao Zhang, Zhizhong Zhang, Yuan Xie, and Yanyun Qu. Multi-memory matching for unsupervised visible-infrared person re-identification. In *ECCV*, 2024.

[38] Jiangming Shi, Xiangbo Yin, Yachao Zhang, Zhizhong Zhang, Yuan Xie, and Yanyun Qu. Learning commonality, divergence and variety for unsupervised visible-infrared person re-identification. *ArXiv*, 2024.

[39] Jiangming Shi, Yachao Zhang, Xiangbo Yin, Yuan Xie, Zhizhong Zhang, Jianping Fan, Zhongchao Shi, and Yanyun Qu. Dual pseudo-labels interactive self-training for semi-supervised visible-infrared person re-identification. In *ICCV*, 2023.

[40] A V Subramanyam, Niranjan Sundararajan, Vibhu Dubey, and Brejesh Lall. Iiitd-20k: Dense captioning for text-image reid. *ArXiv*, 2023.

[41] Hugo Touvron, Louis Martin, Kevin Stone, Peter Albert, Amjad Almahairi, Yasmine Babaei, Nikolay Bashlykov, Soumya Batra, Prajjwal Bhargava, Shruti Bhosale, Dan Bikel, Lukas Blecher, Cristian Canton Ferrer, Moya Chen, Guillem Cucurull, and et. al. Llama 2: Open foundation and fine-tuned chat models. *ArXiv*, 2023.

[42] Laurens van der Maaten and Geoffrey E. Hinton. Visualizing data using t-sne. *JMLR*, 2008.

[43] Guan'an Wang, Tianzhu Zhang, Jian Cheng, Si Liu, Yang Yang, and Zengguang Hou. Rgb-infrared cross-modality person re-identification via joint pixel and feature alignment. In *ICCV*, 2019.

[44] Jianfeng Wang, Zhengyuan Yang, Xiaowei Hu, Linjie Li, Kevin Lin, Zhe Gan, Zicheng Liu, Ce Liu, and Lijuan Wang. Git: A generative image-to-text transformer for vision and language. *ArXiv*, 2022.

[45] Ziyu Wei, Xi Yang, Nannan Wang, and Xinbo Gao. Semi-supervised learning with heterogeneous distribution consistency for visible infrared person re-identification. *IEEE TIP*, 2024.

[46] Ancong Wu, Wei-Shi Zheng, Hong-Xing Yu, Shaogang Gong, and Jianhuang Lai. Rgb-infrared cross-modality person re-identification. In *ICCV*, 2017.

[47] Dongming Wu, Mang Ye, Gaojie Lin, Xin Gao, and Jianbing Shen. Person re-identification by context-aware part attention and multi-head collaborative learning. *IEEE TIFS*, 2022.

[48] Zesen Wu and Mang Ye. Unsupervised visible-infrared person re-identification via progressive graph matching and alternate learning. In *CVPR*, 2023.

[49] Bin Yang, Jun Chen, Cuiqun Chen, and Mang Ye. Dual consistency-constrained learning for unsupervised visible-infrared person re-identification. *IEEE TIFS*, 2024.

[50] Bin Yang, Jun Chen, and Mang Ye. Towards grand unified representation learning for unsupervised visible-infrared person re-identification. In *ICCV*, 2023.

[51] Bin Yang, Jun Chen, and Mang Ye. Shallow-deep collaborative learning for unsupervised visible-infrared person re-identification. In *CVPR*, 2024.

[52] Bin Yang, Mang Ye, Jun Chen, and Zesen Wu. Augmented dual-contrastive aggregation learning for unsupervised visible-infrared person re-identification. In *ACM MM*, 2022.

[53] Mouxing Yang, Peng Huang Zhenyu, Hu, Taihao Li, Jiancheng Lv, and Xi Peng. Learning with twin noisy labels for visible-infrared person re-identification. In *CVPR*, 2022.

[54] Mang Ye, Shuoyi Chen, Chenyue Li, Wei-Shi Zheng, David Crandall, and Bo Du. Transformer for object re-identification: A survey. *ArXiv*, 2024.

[55] Mang Ye, Xiangyuan Lan, and Qingming Leng. Modality-aware collaborative learning for visible thermal person re-identification. In *ACM MM*, 2019.

[56] Mang Ye, Xiangyuan Lan, Jiawei Li, and P C Yuen. Hierarchical discriminative learning for visible thermal person re-identification. In *AAAI*, 2018.

[57] Mang Ye, Weijian Ruan, Bo Du, and Mike Zheng Shou. Channel augmented joint learning for visible-infrared recognition. In *ICCV*, 2021.

[58] Mang Ye, Jianbing Shen, David J. Crandall, Ling Shao, and Jiebo Luo. Dynamic dual-attentive aggregation learning for visible-infrared person re-identification. In *ECCV*, 2020.

[59] Mang Ye, Jianbing Shen, Gaojie Lin, Tao Xiang, Ling Shao, and Steven C. H. Hoi. Deep learning for person re-identification: A survey and outlook. *IEEE TPAMI*, 2022.

[60] Mang Ye, Zesen Wu, Cuiqun Chen, and Bo Du. Channel augmentation for visible-infrared re-identification. *IEEE TPAMI*, 2024.

[61] Xiangbo Yin, Jiangming Shi, Yachao Zhang, Yang Lu, Zhizhong Zhang, Yuan Xie, and Yanyun Qu. Robust pseudo-label learning with neighbor relation for unsupervised visible-infrared person re-identification. *ArXiv*, 2024.

[62] Zhiming Luo Yansong Qu Rongrong Ji Min Jiang Yunpeng Gong, Zhun Zhong. Cross-modality perturbation synergy attack for person re-identification. *ArXiv*, 2024.

[63] Xuanmeng Zhang, Minyue Jiang, Zhedong Zheng, Xiaoping Tan, Errui Ding, and Yi Yang. Understanding image retrieval re-ranking: A graph neural network perspective. *ArXiv*, 2020.

[64] Yiyuan Zhang, Sanyuan Zhao, Yuhao Kang, and Jianbing Shen. Modality synergy complement learning with cascaded aggregation for visible-infrared person re-identification. In *ECCV*, 2022.

[65] Yukang Zhang and Hanzi Wang. Diverse embedding expansion network and low-light cross-modality benchmark for visible-infrared person re-identification. In *CVPR*, 2023.

[66] Yukang Zhang, Yan Yan, Yang Lu, and Hanzi Wang. Towards a unified middle modality learning for visible-infrared person re-identification. In *ACM MM*, 2021.

[67] Lianmin Zheng, Wei-Lin Chiang, Ying Sheng, Siyuan Zhuang, Zhanghao Wu, Yonghao Zhuang, Zi Lin, Zhuohan Li, Dacheng Li, Eric Xing, Hao Zhang, Joseph E Gonzalez, and Ion Stoica. Judging llm-as-a-judge with mt-bench and chatbot arena. In *NeurIPS*, 2023.

[68] Zhihui Zhu, Xinyang Jiang, Feng Zheng, Xiaowei Guo, Feiyue Huang, Weishi Zheng, and Xing Sun. Viewpoint-aware loss with angular regularization for person re-identification. *ArXiv*, 2019.

[69] Chang Zou, Zeqi Chen, Zhichao Cui, Yuehu Liu, and Chi Zhang. Discrepant and multi-instance proxies for unsupervised person re-identification. In *ICCV*, 2023.

## A Details of Expanded Datasets

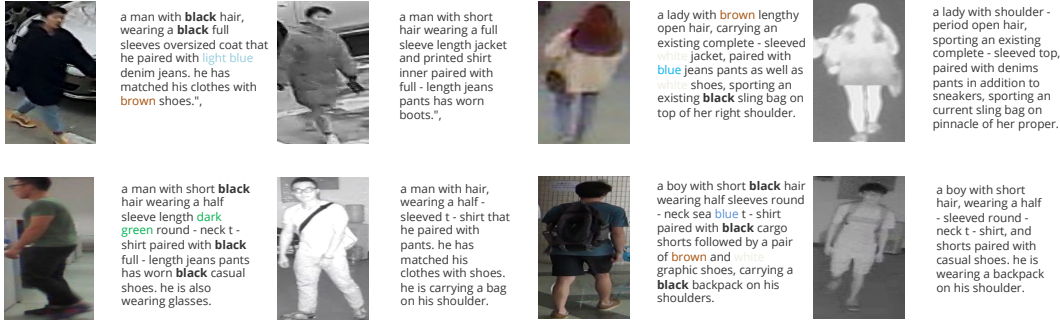

Figure 5: Visualization of the data samples selected from the expanded three datasets.

Table 4: Dataset statistics

| Datasets | #ID | #RGB | #IR | #Text |
|---|---|---|---|---|
| Tri-LLCM | 1064 | 25626 | 21141 | 46767 |
| Tri-RegDB | 412 | 4120 | 4120 | 8240 |
| Tri-SYSU-MM01 | 491 | 30071 | 15792 | 45863 |

All the fine-tuning process of VLMs can be found in documentations from huggingface: `https://huggingface.co/docs/transformers/main/en/tasks/image_captioning`.

The generator model we use refers to the official implementation released in huggingface: `https://huggingface.co/Salesforce/blip-image-captioning-large`.

## B Implementation Details

We implement our framework in PyTorch [32] utilizing a single NVIDIA RTX 3090 GPU for training. For visual backbone training, it takes about 9GB memory for training and about 3GB memory for testing, about 9 hours are needed for training on Tri-SYSU-MM01 and Tri-LLCM, about 1 hour for smaller Tri-RegDB. For incremental fine-tuing, it takes about 5GB memory for training and about 3GB memory for testing, about 1 hour are needed for fine-tuning on Tri-SYSU-MM01 and Tri-LLCM, about 10 miniutes for smaller Tri-RegDB. Each batch consists of 8 identities, with each identity containing 4 visible images, 4 infrared images, 4 text descriptions generated from visible images, and 4 text descriptions generated from infrared images. All input images are resized to $3 \times 288 \times 144$, with full augmentation strategy the same as CAJ [57]. All text descriptions are generated by two modality-specialized fine-tuned VLMs and augmented by the proposed LLM rephrasing augmentation with a probability of 0.5, here we use vicuna-7b [67] as our LLM model, use Blip [24] as our VLM model, whose tuning process can be found in Sec. 3.1. We employ a dual-stream resnet50 model [58] pre-trained on ImageNet [36] as the visual backbone and a transformer model with parameters derived from CLIP [34] as the textual backbone. For incrementally fine-tuning our TVI-LFM, at first, we should get an available well-trained visual backbone. Here we utilize the augmentation method [57] to train the visual backbone for 120 epochs by cross-entropy loss and weighted regularized triplet loss [59], finally get the well-trained visual backbone. Then we integrate the well-trained VI-ReID model and fine-tune the text encoder from CLIP [34] and a simple ReID bottleneck [29] applied for each feature for 20 epochs. We use the Adam [17] for optimization. For the Tri-SYSU-MM01 and Tri-LLCM datasets, in both visual and textual parts, the learning rate is set to 3.5e-4 and the weight decay to 5e-4. For the Tri-RegDB dataset, the learning rate for the visual part is 2e-3 with weight decay of 5e-4, and for the textual part, the learning rate is 1e-5 with weight decay of 4e-5. The learning rate rises up to the initial value by a linear warm-up scheme for the first 10 epochs, then decays by a linear scheme with a decay-factor of 0.1 at the milestones of 40, 60, and 100 epochs.

## C Retrieve Result Examples w/wo Text

| Method | Target Description | Generated Filter | IR | Results |
|---|---|---|---|---|
| VI-ReID Backbone | N/A | N/A |  |  |
| Baseline | a short **black hair** boy wearing short sleeves **red t - shirt** with grey jogger pants, paired with **green shoes…** | N/A |  |  |
| Ours | a short **black hair** boy wearing short sleeves **red t - shirt** with grey jogger pants, paired with **green shoes…** | a short hair boy wearing short sleeves t - shirt with jogger pants, paired with shoes… |  |  |

| Method | Target Description | Generated Filter | IR | Results |
|---|---|---|---|---|
| VI-ReID Backbone | N/A | N/A |  |  |
| Baseline | a man with **black hair**, wearing a **green half sleeves t - shirt** that he paired with **dark black** colour … | N/A |  |  |
| Ours | a man with **black hair**, wearing a **green half sleeves t - shirt** that he paired with **dark black** colour … | a man with hair, wearing a half sleeves t - shirt that he paired with … |  | 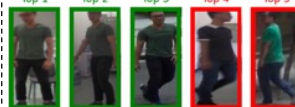 |

Figure 6: Visualization of the rank-5 retrieval results obtained by the VI-ReID backbone, the baseline, and our method on the proposed Tri-SYSU-MM01.

The VI-ReID backbone and the baseline still includes misidentifications. But our method fully leverages complementary information from textual data, significantly enhancing retrieval performance through semantic filtered fusion.

## D Assets Details

This section provides the necessary details for the data assets utilized in our research: SYSU-MM01, LLCM, and RegDB.

- **SYSU-MM01** [46]
  - *Source and Citation*: The SYSU-MM01 dataset was created by researchers at Sun Yat-sen University (SYSU). Ancong Wu, et al. "RGB-IR Person Re-Identification by Cross-Modality Similarity Preservation" (2020) is the seminal paper associated with this dataset.
  - *data splits*: The training set contains 22,258 visible images and 11,909 infrared images of 395 identities. The testing set contains 96 identities, with 3,803 infrared images for query and 301 (single-shot) randomly selected visible images as the gallery set.
  - *URL*: The dataset can be accessed through a GitHub repository: `https://github.com/wuancong/SYSU-MM01` , where users must agree to the data release agreement.
  - *License*: We cannot find out the license SYSU-MM01 uses, but the author requires signing the usage agreement notice and contact him through e-mail to get the dataset. The detailed usage agreement refers to the github url mentioned above.

- **LLCM** [65]
  - *Source and Citation*: The LLCM dataset was introduced by researchers from Xiamen University. Yukang Zhang and Hanzi Wang's paper "Diverse Embedding Expansion Network and Low-Light Cross-Modality Benchmark for Visible-Infrared Person Re-identification" (2023) discusses this dataset.
  - *data splits*: The training set contains 30,921 images of 713 identities, and the test set contains 13,909 images of 351 identities.
  - *URL*: The dataset is available on GitHub `https://github.com/ZYK100/LLCM`.
  - *License*: CC-BY 4.0
  - *Code*: We use its code for feature visualization.
- **RegDB** [31]
  - *Source and Citation*: The RegDB dataset was developed at Dongguk University from the paper named "Person Recognition System Based on a Combination of Body Images from Visible Light and Thermal Cameras".
  - *data splits*: The training set contains 206 identities and the testing set contains 206 identities. There are 10 visible images and 10 infrared images for each person.
  - *URL*: We can only find the paper's doi `https://doi.org/10.3390/s17030605`
  - *License*: CC-BY 4.0

# E  Discussion of Incremental Fine-tuning Strategy

Table 5: The impact of incrementally fine-tuning the framework based on a frozen, well-trained visual backbone versus training from scratch is evaluated for two scenarios: infrared query ($I \rightarrow R$) and fusion query ($I + T \rightarrow R$) on the performance of **Tri-SYSU-MM01** and **Tri-LLCM**. To specifically analyze the influence on visual perception capability and fusion feature modeling, we **exclude** the **MER** strategy from the fusion query to eliminate the effect of combining original features from both the infrared modality and text modality.

| $I \rightarrow R$ | Tri-SYSU-MM01 | | | Tri-LLCM | | |
|---|---|---|---|---|---|---|
| | R1 | mAP | mINP | R1 | mAP | mINP |
| VI-ReID Backbone | 69.89 | 66.74 | 53.34 | 53.53 | 59.77 | 56.40 |
| From Scratch | 64.46↓5.43 | 61.31↓5.43 | 46.94↓6.40 | 49.29↓4.24 | 55.78↓3.99 | 52.12↓4.28 |

| $I + T \rightarrow R$ | Tri-SYSU-MM01 | | | Tri-LLCM | | |
|---|---|---|---|---|---|---|
| | R1 | mAP | mINP | R1 | mAP | mINP |
| TVI-LFM | **84.17** | **80.72** | **70.02** | **57.13** | **64.06** | **60.72** |
| From Scratch | 84.03↓0.14 | 79.85↓0.87 | 68.06↓1.97 | 55.47↓1.66 | 62.23↓1.83 | 58.86↓1.86 |

To optimize the whole framework, we first train a simple VI-ReID backbone, then incrementally fine-tune the VLM textual encoder based on the well-trained frozen backbone to inherit its visual perception capability and integrate text information for infrared modality compensation. If we train the whole framework from scratch, as shown in the Table 5 above, the performance of the VI-ReID backbone suddenly declines by 5.43% and 4.24% in Rank-1 in the two datasets respectively, indicating the loss of visual perception capability, thereby the performance textually enhanced task ($I + T \rightarrow R$) is also affected, with a decline of 0.14% Rank-1 in Tri-SYSU-MM01 and 1.66% Rank-1 in Tri-LLCM. This demonstrates the importance of incrementally fine-tuning strategy, which avoids the potential performance influence caused by conflicts of modeling visual features and textual enhanced infrared features optimization.

# F  Broader Impacts

Our TVI-LFM framework offers significant advancements in urban security by enhancing person re-identification in low-light conditions, boosting surveillance effectiveness. It automates text generation from IR and RGB images, reducing annotation workload and improving text robustness, aiding multi-modal research and smart security system development. However, it's crucial to address

environmental impact concerns related to large models' energy consumption and the privacy risks associated with re-identification technology. Governments and regulatory bodies must enact stringent regulations to prevent misuse and ensure identification accuracy to avoid societal disruptions.

